# Scaling of Probability-Based Optimization Algorithms

**J. L. Shapiro**

Department of Computer Science University of Manchester
Manchester, M13 9PL U.K. *jls@cs.man.ac.uk*

## Abstract

Population-based Incremental Learning is shown require very sensitive scaling of its learning rate. The learning rate must scale with the system size in a problem-dependent way. This is shown in two problems: the needle-in-a haystack, in which the learning rate must vanish exponentially in the system size, and in a smooth function in which the learning rate must vanish like the square root of the system size. Two methods are proposed for removing this sensitivity. A learning dynamics which obeys detailed balance is shown to give consistent performance over the entire range of learning rates. An analog of mutation is shown to require a learning rate which scales as the inverse system size, but is problem independent.

## 1   Introduction

There has been much recent work using probability models to search in optimization problems. The probability model generates candidate solutions to the optimization problem. It is updated so that the solutions generated should improve over time. Usually, the probability model is a parameterized graphical model, and updating the model involves changing the parameters and possibly the structure of the model. The general scheme works as follows,

- Initialize the model to some prior (e.g. a uniform distribution);
- Repeat
    - **Sampling step**: generate a data set by sampling from the probability model;
    - **Testing step**: test the data as solutions to the problem;
    - **Selection step**: create a improved data set by selecting the better solutions and removing the worse ones;
    - **Learning step**: create a new probability model from the old model and the improved data set (e.g. as a mixture of the old model and the most likely model given the improved data set);
- until (stopping criterion met)

Different algorithms are largely distinguished by the class of probability models used. For reviews of the approach including the different graphical models which

have been used, see [3, 6]. These algorithms have been called Estimation of Distribution Algorithms (EDA); I will use that term here.

EDAs are related to genetic algorithms; instead of evolving a population, a generative model which produces the population at each generation is evolved. A motivation for using EDAs instead of GAs is that is that in EDAs the structure of the graphical model corresponds to the form of the crossover operator in GAs (in the sense that a given graph will produce data whose probability will not change much under a particular crossover operator). If the EDA can learn the structure of the graph, it removes the need to set the crossover operator by hand (but see [2] for evidence against this).

In this paper, a very simple EDA is considered on very simple problems. It is shown that the algorithm is extremely sensitive to the value of learning rate. The learning rate must vanish with the system size in a problem dependent way, and for some problems it has to vanish exponentially fast. Two correctives measures are considered: a new learning rule which obeys detailed balance in the space of parameters, and an operator analogous to mutation which has been proposed previously.

## 2    The Standard PBIL Algorithm

The simplest example of a EDA is Population-based Incremental Learning (PBIL) which was introduced by Baluja [1]. PBIL uses a probability model which is a product of independent probabilities for each component of the binary search space. Let $x_i$ denote the $i$th component of $\vec{x}$, an $L$-component binary vector which is a state of the search space. The probability model is defined by the $L$-component vector of parameters $\vec{\gamma}(t)$, where $\gamma_i(t)$ denotes the probability that $x_i = 1$ at time $t$.

The algorithm works as follows,

- Initialize $\gamma_i(0) = 1/2$ for all $i$;
- Repeat
    - Generate a population of $N$ strings by sampling from the binomial distribution defined by $\vec{\gamma}(t)$.
    - Find the best string in the population $\vec{x^*}$.
    - Update the parameters $\gamma_i(t+1) = \gamma_i(t) + \alpha[x_i^* - \gamma_i(t)]$ for all $i$.
- until (stopping criterion met)

The algorithm has only two parameters, the size of the population $N$ and the learning parameter $\alpha$.

## 3    The sensitivity of PBIL to the learning rate

### 3.1    PBIL on a flat landscape

The source of sensitivity of PBIL to the learning rate lies in its behavior on a flat landscape. In this case all vectors are equally fit, so the "best" vector $\vec{x^*}$ is a random vector and its expected value is

$$\langle x_i^* \rangle = \gamma_i, \tag{1}$$

(where $\langle \cdot \rangle$ denotes the expectation operator) Thus, the parameters remain unchanged on average. In any individual run, however, the parameters converge

rapidly to one of the corners of the hypercube. As the parameters deviate from 1/2 they will move towards a corner of the hypercube. Then the population generated will be biased towards that corner, which will move the parameters closer yet to that corner, etc. All of the corners of the hypercube are attractors which, although never reached, are increasingly attractive with increasing proximity. Let us call this phenomenon *drift*. (In population genetics, the term drift refers to the loss of genetic diversity due to finite population sampling. It is in analogy to this that the term is used here.)

Consider the average distance between the parameters and 1/2,

$$D(t) \equiv \frac{1}{L} \sum_i \left( \frac{1}{2} - \gamma_i(t) \right)^2. \tag{2}$$

Solving this reveals that on average this converges to 1/4 with a characteristic time

$$\tau = -1/\log(1 - \alpha^2) \approx 1/\alpha^2 \text{ for } \alpha \approx 0. \tag{3}$$

The rate of search on any other search space will have to compete with drift.

## 3.2 PBIL and the needle-in-the haystack problem

As a simple example of the interplay between drift and directed search, consider the so-called needle-in-a-haystack problem. Here the fitness of all strings is 0 except for one special string (the "needle") which has a fitness of 1. Assume it is the string of all 1's. It is shown here that PBIL will only find the needle if $\alpha$ is exponentially small, and is inefficient at finding the needle when compared to random search.

Consider the probability of finding the needle at time $t$, denoted $\Omega(t) = \prod_{i=1}^{L} \gamma_i(t)$. Consider times shorter than $T$ where $T$ is long enough that the needle may be found multiple times, but $\alpha^2 T \to 0$ as $L \to \infty$. It will be shown for small $\alpha$ that when the needle is not found (during drift), $\Omega$ decreases by an amount $\alpha^2 L\Omega/2$, whereas when the needle is found, $\Omega$ increases by the amount $\alpha L\Omega$. Since initially, the former happens at a rate $2^L$ times greater than the latter, $\alpha$ must be less than $2^{-(L-1)}$ for the system to move towards the hypercube corner near the optimum, rather than towards a random corner.

When the needle is not found, the mean of $\Omega(t)$ is invariant, $\langle \Omega(t+1) \rangle = \Omega(t)$. However, this is misleading, because $\Omega$ is not a self-averaging quantity; its mean is affected by exponentially unlikely events which have an exponentially big effect. A more robust measure of the size of $\Omega(t)$ is the exponentiated mean of the log of $\Omega(t)$. This will be denoted by $[\![\Omega]\!] \equiv \exp\langle \log \Omega \rangle$. This is the appropriate measure of the central tendency of a distribution which is approximately log-normal [4], as is expected of $\Omega(t)$ early in the dynamics, since the log of $\Omega$ is the sum of approximately independent quantities.

The recursion for $\Omega$ expanded to second order in $\alpha$ obeys

$$[\![\Omega(t+1)]\!] = \begin{cases} [\![\Omega(t)]\!] \left[1 - \frac{1}{2}\alpha^2 L\right]; & \text{needle not found} \\ [\![\Omega(t)]\!] \left[1 + \alpha L + \frac{1}{2}\alpha^2 L(L-1)\right]; & \text{needle found.} \end{cases} \tag{4}$$

In these equations, $\gamma_i(t)$ has also been expanded around 1/2.

Since the needle will be found with probability $\Omega(t)$ and not found with probability $1 - \Omega(t)$, the recursion averages to,

$$[\![\Omega(t+1)]\!] = [\![\Omega(t)]\!] \left(1 - \frac{1}{2}\alpha^2 L\right) + [\![\Omega(t)]\!]^2 \left[\alpha L - \frac{1}{2}\alpha^2 L(L+1)\right]. \tag{5}$$

The second term actually averages to $[\![\Omega(t)]\!] \langle\Omega(t)\rangle$, but the difference between $\langle\Omega\rangle$ and $[\![\Omega]\!]$ is of order $\alpha$, and can be ignored.

Equation (5) has a stable fixed point at 0 and an unstable fixed point at $\alpha/2 + O(\alpha^2 L)$. If the initial value of $\Omega(0)$ is less than the unstable fixed point, $\Omega$ will decay to zero. If $\Omega(0)$ is greater than the unstable fixed point, $\Omega$ will grow. The initial value is $\Omega(0) = 2^{-L}$, so the condition for the likelihood of finding the needle to increase rather than decrease is $\alpha < 2^{-(L-1)}$.

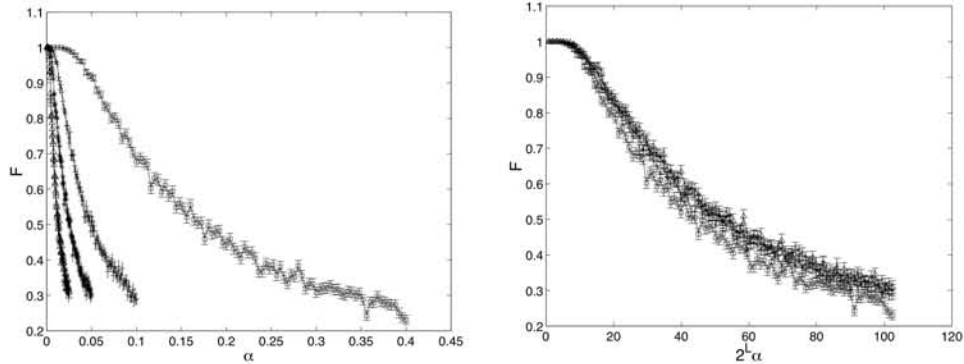

Figure 1: Simulations on PBIL on needle-in-a-haystack problem for $L = 8, 10, 11, 12$ (respectively $\square, +, *, \triangle$). The algorithm is run until no parameters are between 0.05 and 0.95, and averaged over 1000 runs. **Left:** Fitness of best population member at convergence versus $\alpha$. The non-robustness of the algorithm is clear; as $L$ increases, $\alpha$ must be very finely set to a very small value to find the optimum. **Right:** As previous, but with $\alpha$ scaled by $2^L$. The data approximately collapses, which shows that as $L$ increases, $\alpha$ must decrease like $2^{-L}$ to get the same performance.

Figure 1 shows simulations of PBIL on the needle-in-a-haystack problem. These confirm the predictions made above, the optimum is found only if $\alpha$ is smaller than a constant times $2^L$. The algorithm is inefficient because it requires such small $\alpha$; convergence to the optimum scales like $4^L$. This is because the rate of convergence to the optimum goes like $\Omega\alpha$, both of which are $O(2^{-L})$.

### 3.3  PBIL and functions of unitation

One might think that the needle-in-the-haystack problem is hard in a special way, and results on this problem are not relevant to other problems. This is not be true, because even smooth functions have flat subspaces in high dimensions. To see this, consider any continuous, monotonic function of unitation $u$, where $u = \frac{1}{L}\sum_i^L x_i$, the number of 1's in the vector. Assume the the optimum occurs when all components are 1.

The parameters $\vec{\gamma}$ can be decomposed into components parallel and perpendicular to the optimum. Movement along the perpendicular direction is neutral, Only movement towards or away from the optimum changes the fitness. The random strings generated at the start of the algorithm are almost entirely perpendicular to the global optimum, projecting only an amount of order $1/\sqrt{L}$ towards the optimum.

Thus, the situation is like that of the needle-in-a-haystack problem. The perpendicular direction is flat, so there is convergence towards an arbitrary hypercube corner

with a drift rate,

$$\tau_{\perp} \sim \alpha^2. \tag{6}$$

from equation (3). Movement towards the global optimum occurs at a rate,

$$\tau_{\|} \sim \frac{\alpha}{\sqrt{L}}. \tag{7}$$

Thus, $\alpha$ must be small compared to $1/\sqrt{L}$ for movement towards the global optimum to win.

A rough argument can be used to show how the fitness in the final population depends on $\alpha$. Making use of the fact that when $N$ random variables are drawn from a Gaussian distribution with mean $m$ and variance $\sigma^2$, the expected largest value drawn is $m + \sqrt{2\sigma^2 \log(N)}$ for large $N$ (see, for example, [7]), the Gaussian approximation to the binomial distribution, and approximating the expectation of the square root as the square root of the expectation yields,

$$\langle u(t+1) \rangle = \langle u(t) \rangle + \alpha \sqrt{2 \langle v(t) \rangle \log(N)}, \tag{8}$$

where $v(t)$ is the variance in probability distribution, $v(t) = \frac{1}{L^2} \sum_i \gamma_i(t)[1 - \gamma_i(t)]$. Assuming that the convergence of the variance is primarily due to the convergence on the flat subspace, this can be solved as,

$$\langle u(\infty) \rangle \approx \frac{1}{2} + \frac{\sqrt{\log(N)}}{\alpha \sqrt{2L}}. \tag{9}$$

The equation must break down when the fitness approaches one, which is where the Gaussian approximation to the binomial breaks down.

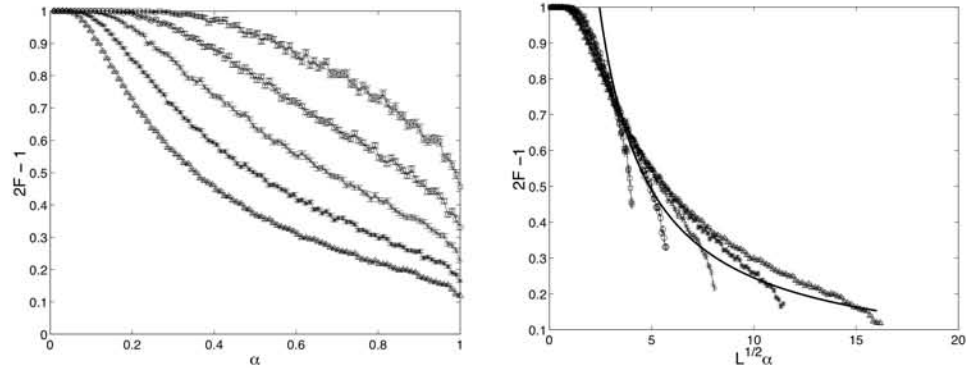

Figure 2: Simulations on PBIL on the unitation function for $L = 16, 32, 64, 128, 256$ (respectively $\square, \circ, +, *, \triangle$). The algorithm is run until all parameters are closer to 1 or 0 than 0.05, and averaged over 100 runs. **Left:** Fitness of best population member at convergence versus $\alpha$. The fitness is scaled so that the global optimum has fitness 1 and the expected fitness of a random string is 0. As $L$ increases, $\alpha$ must be set to a decreasing value to find the optimum. **Right:** As previous, but with $\alpha$ scaled by $\sqrt{L}$. The data approximately collapses, which shows that as $L$ increases, $\alpha$ must decrease like $\sqrt{L}$ to get the same performance. The smooth curve shows equation (9).

Simulations of PBIL on the unitation function confirm these predictions. PBIL fails to converge to the global optimum unless $\alpha$ is small compared to $1/\sqrt{L}$. Figure 2 shows the scaling of fitness at convergence with $\alpha\sqrt{L}$, and compares simulations with equation (9).

# 4 Corrective 1 — Detailed Balance PBIL

One view of the problem is that it is due to the fact that the learning dynamics does not obey *detailed balance*. Even on a flat space, the rate of movement of the parameters $\gamma_i$ away from $1/2$ is greater than the movement back. It is well-known that a Markov process on variables $x$ will converge to a desired equilibrium distribution $\pi(x)$ if the transition probabilities obey the detailed balance conditions,

$$w(x'|x)\pi(x) = w(x|x')\pi(x'), \tag{10}$$

where $w(x'|x)$ is the probability of generating $x'$ from $x$. Thus, any search algorithm searching on a flat space should have dynamics which obeys,

$$w(x'|x) = w(x|x'), \tag{11}$$

and PBIL does not obey this. Perhaps the sensitive dependence on $\alpha$ would be removed if it did.

There is a difficulty in modifying the dynamics of PBIL to satisfy detailed balance, however. PBIL visits a set of points which varies from run to run, and (almost) never revisits points. This can be fixed by constraining the parameters to lie on a lattice. Then the dynamics can be altered to enforce detailed balance.

Define the allowed parameters in terms of a set of integers $n_i$. The relationship between them is.

$$\gamma_i = \begin{cases} 1 - \frac{1}{2}(1-\alpha)^{n_i}, & n_i > 0; \\ \frac{1}{2}(1-\alpha)^{|n_i|}, & n_i < 0; \\ \frac{1}{2}, & n_i = 0. \end{cases} \tag{12}$$

Learning dynamics now consists of incrementing and decrementing the $n_i$'s by 1; when $x_i^* = 1(0)$ $n_i$ is incremented (decremented).

Transforming variables via equation (12), the uniform distribution in $\gamma$ becomes in $n$,

$$P(n) = \frac{\alpha}{2-\alpha}(1-\alpha)^{|n|}. \tag{13}$$

### 4.0.1 Detailed balance by rejection sampling

One of the easiest methods for sampling from a distribution is to use the rejection method. In this, one has $g(x'|x)$ as a proposal distribution; it is the probability of proposing the value $x'$ from $x$. Then, $A(x'|x)$ is the probability of accepting this change. Detailed balance condition becomes

$$g(x'|x)A(x'|x)\pi(x) = g(x|x')A(x|x')\pi(x'). \tag{14}$$

For example, the well-known Metropolis-Hasting algorithm has

$$A(x'|x) = \min\left(1, \frac{\pi(x')g(x|x')}{\pi(x)g(x'|x)}\right). \tag{15}$$

The analogous equations for PBIL on the lattice are,

$$A(n+1|n) = \min\left[\frac{1-\gamma(n+1)}{\gamma(n)}(1-\alpha), 1\right] \tag{16}$$

$$A(n-1|n) = \min\left[\frac{\gamma(n-1)}{1-\gamma(n)}(1-\alpha), 1\right]. \tag{17}$$

In applying the acceptance formula, each component is treated independently. Thus, moves can be accepted on some components and not on others.

#### 4.0.2 Results

Detailed Balance PBIL requires no special tuning of parameters, at least when applied to the two problems of the opening sections. For the needle-in-a-haystack, simulations were performed for 100 values of $\alpha$ between 0 and 0.4 equally spaced for $L = 8, 9, 10, 11, 12$; 1000 trials of each, population size 20, with the same convergence criterion as before, simulation halts when all $\gamma_i$'s are less than 0.05 or greater than 0.95. On none of those simulations did the algorithm fail to contain the global optimum in the final population.

For the function of unitation, Detailed Balance PBIL appears to always find the optimum if run long enough. Stopping it when all parameters fell outside the range $(0.05, 0.95)$, the algorithm did not always find the global optimum. It produced an average fitness within 1% of the optimum for $\alpha$ between 0.1 and 0.4 and $L = 32, 64, 128, 256$ over a 100 trials, but for learning rates below 0.1 and $L = 256$ the average fitness fell as low as 4% below optimum. However, this is much improved over standard PBIL (see figure 2) where the average fitness fell to 60% below the optimum in that range.

## 5 Corrective 2 — Probabilistic mutation

Another approach to control drift is to add an operator analogous to mutation in GAs. Mutation has the property that when repeatedly applied, it converges to a random data set. Muhlenbein [5] has proposed that the analogous operator EDAs estimates frequencies biased towards a random guess. Suppose $\tilde{\gamma}_i$ is the fraction of 1's at site $i$. Then, the appropriate estimate of the probability of a 1 at site $i$ is

$$\gamma_i = \frac{\tilde{\gamma}_i + m}{1 + 2m},\tag{18}$$

where $m$ is a mutation-like parameter. This will be recognized as the maximum posterior estimate of the binomial distribution using as the prior a $\beta$-distribution with both parameters equal to $mN + 1$; the prior biases the estimate towards $1/2$. This can be applied to PBIL by using the following learning rule,

$$\gamma_i(t+1) = \frac{\gamma_i(t) + \alpha\left[x_i^* - \gamma_i(t)\right] + m}{1 + 2m}.\tag{19}$$

With $m = 0$ it gives the usual PBIL rule; when repeatedly applied on a flat space it converges to $1/2$.

Unlike Detailed Balance PBIL, this approach does required special scaling of the learning rate, but the scaling is more benign than in standard PBIL and is problem independent. It is determined from three considerations. First, mutation must be large enough to counteract the effects of drift towards random corners of the hypercube. Thus, the fixed point of the average distance to $1/2$, $\langle D(t+1)\rangle$ defined in equation (2), must be sufficiently close to zero. Second, mutation must be small enough that it does not interfere with movement towards the parameters near the optimum when the optimum is found. Thus, the fixed point of equation (19) must be sufficiently close to 0 or 1. Finally, a sample of size $N$ sampled from the fixed point distribution near the hypercube corner containing the optimum should contain the optimum with a reasonable probability (say greater than $1 - e^{-1}$). Putting these considerations together yields,

$$\frac{\log N}{L} >> \frac{m}{\alpha} >> \frac{\alpha}{4}.\tag{20}$$

## 5.1 Results

To satisfy the conditions in equation 20, the mutation rate was set to $m \propto \alpha^2$, and $\alpha$ was constrained to be smaller than $\log(N)/L$. For the needle-in-a-haystack, the algorithm behaved like Detailed Balance PBIL. It never failed to find the optimum for the needle-in-a-haystack problems for the sizes given previously. For the functions of unitation, no improvement over standard PBIL is expected, since the scaling using mutation is worse, requiring $\alpha < 1/L$ rather than $\alpha < 1/\sqrt{L}$. However, with tuning of the mutation rate, the range of $\alpha$'s with which the optimum was always found could be increased over standard PBIL.

## 6 Conclusions

The learning rate of PBIL has to be very small for the algorithm to work, and unpredictably so as it depends upon the problem size in a problem dependent way. This was shown in two very simple examples. Detailed balance fixed the problem dramatically in the two cases studied. Using detailed balance, the algorithm consistently finds the optimum over the entire range of learning rates. Mutation also fixed the problem when the parameters were chosen to satisfy a problem-independent set of inequalities.

The phenomenon studied here could hold in any EDA, because for any type of model, the probability is high of generating a population which reinforces the move just made. On the other hand, more complex models have many more parameters, and also have more sources of variability, so the issue may be less important. It would be interesting to learn how important this sensitivity is in EDAs using complex graphical models.

Of the proposed correctives, detailed balance will be more difficult to generalize to models in which the structure is learned. It requires an understanding of algorithm's dynamics on a flat space, which may be very difficult to find in those cases. The mutation-type operator will easier to generalize, because it only requires a bias towards a random distribution. However, the appropriate setting of the parameters may be difficult to ascertain.

## References

[1] S. Baluja. Population-based incremental learning: A method for integrating genetic search based function optimization and competive learning. Technical Report CMU-CS-94-163, Computer Science Department, Carnegie Mellon University, 1994.

[2] A. Johnson and J. L. Shapiro. The importance of selection mechanisms in distribution estimation algorithms. In *Proceedings of the $5^{th}$ International Conference on Artificial Evolution AE01*, 2001.

[3] P. Larrañaga and J. A. Lozano. *Estimation of Distribution Algorithms, A New Tool for Evolutionary Computation*. Kluwer Academic Publishers, 2001.

[4] Eckhard Limpert, Werner A. Stahel, and Markus Abbt. Log-normal distributions across the sciences: Keys and clues. *BioScience*, 51(5):341–352, 2001.

[5] H. Mühlenbein. The equation for response to selection and its use for prediction. *Evolutionary Computation*, 5(3):303–346, 1997.

[6] M. Pelikan, D. E. Goldberg, and F. Lobo. A survey of optimization by building and using probabilistic models. Technical report, University of Illinois at Urbana-Champaign, Illinois Genetic Algorithms Laboratory, 1999.

[7] Jonathan L. Shapiro and Adam Prügel-Bennett. Maximum entropy analysis of genetic algorithm operators. *Lecture Notes in Computer Science*, 993:14–24, 1995.
